# On the (Non-)existence of Convex, Calibrated Surrogate Losses for Ranking

**Clément Calauzènes, Nicolas Usunier, Patrick Gallinari**
LIP6 - UPMC
4 place Jussieu, 75005 Paris, France
`firstname.lastname@lip6.fr`

## Abstract

We study surrogate losses for learning to rank, in a framework where the rankings are induced by scores and the task is to learn the scoring function. We focus on the calibration of surrogate losses with respect to a ranking evaluation metric, where the calibration is equivalent to the guarantee that near-optimal values of the surrogate risk imply near-optimal values of the risk defined by the evaluation metric. We prove that if a surrogate loss is a convex function of the scores, then it is not calibrated with respect to two evaluation metrics widely used for search engine evaluation, namely the Average Precision and the Expected Reciprocal Rank. We also show that such convex surrogate losses cannot be calibrated with respect to the Pairwise Disagreement, an evaluation metric used when learning from pairwise preferences. Our results cast lights on the intrinsic difficulty of some ranking problems, as well as on the limitations of learning-to-rank algorithms based on the minimization of a convex surrogate risk.

## 1 Introduction

A surrogate loss is a loss function used as a substitute for the true quality measure during training in order to ease the optimization of the empirical risk. The hinge loss or the exponential loss, which are used in Support Vector Machines or AdaBoost as convex upper bounds of the classification error, are well-known examples of surrogate losses for binary classification. In this paper, we study surrogate losses for learning to rank, in a context where a set of items should be ranked given an input query and where the ranking is obtained by sorting the items according to predicted numerical scores. This work is motivated by the intensive research that has recently been carried out on machine learning approaches to improve the quality of search engine results, and more specifically on the design of surrogate losses that lead to high quality rankings (see [16] for a review).

Considering algorithms for learning to rank on the axis of scalability, there are first algorithms that are designed for small-scale datasets only and that directly solve the NP-hard problem [5] without using any surrogate loss; after them come algorithms that use a surrogate loss chosen as a non-convex but continuous and (almost everywhere) differentiable approximation of the evaluation metric [3, 21, 10], and finally algorithms that use a convex surrogate loss. Most algorithms for learning to rank fall into the latter category, including the reference algorithms RankBoost [12] and Ranking SVMs [14, 4] or the regression approach of [8], because convex surrogate losses lead to optimization problems that can be solved efficiently while non-convex approaches may require intensive computations to find a good local optimum. The disadvantage of convex surrogate losses is that they cannot closely approximate the evaluation metrics on the whole prediction space. However, as more examples are available and smaller values of the surrogate risk are achieved, the only region of interest becomes that of near-optimal predictions. It is thus possible that the minimization of the surrogate risk provably leads to optimal predictions according to the risk defined by the evaluation measure. In that case, the surrogate loss is said to be calibrated with respect to the evaluation metric.

The calibration of surrogate losses has been extensively studied for various classification settings [1, 26, 27, 18, 19] and for AUC optimization [7, 15]. For each of these tasks, many usual convex losses are calibrated with respect to the natural evaluation metric. In the context of learning to rank for search engines, several families of convex losses are calibrated with respect to the Discounted Cumulative Gain (DCG) and its variants [8, 2, 17]. However, other metrics than the DCG are often used as reference for the evaluation of ranked results, such as the Average Precision (AP), used in past TREC competitions [22], the Expected Reciprocal Rank (ERR), used the Yahoo! Learning to Rank Challenge [6], or the Pairwise Disagreement (PD), used when learning from pairwise preferences. And despite the multiplicity of convex losses that have been proposed for ranking, none of them was proved to be calibrated with respect to any of these three metrics. This lead us to the question of whether convex losses can be calibrated with respect to the AP, the ERR, or the PD.

Our main contribution is a definitive and negative answer to that question. We prove that if a surrogate loss is convex, then it *cannot* be calibrated with respect to any of the AP, the ERR or the PD. Thus, if one of these metrics should be optimized, the price to pay for the computational advantage of convex losses is an inconsistent learning procedure, which may converge to non-optimal predictions as the number of examples increases.

Our result generalizes previous works on non-calibration. First, Duchi et al. [11] showed that many convex losses based on pairwise comparisons, such as those of RankBoost [12] or Ranking SVMs [14, 4], are not calibrated with respect to the PD. Secondly, Buffoni et al. [2] showed that specific convex losses, called order-preserving, are not calibrated with respect to the AP or the ERR, even though these losses are calibrated with respect to (any variant of) the DCG. Our result is stronger than those because we do not make any assumption on the exact structure of the loss; our approach as a whole is also more general because it directly applies to the three evaluation metrics (AP, ERR and PD). Finally, Duchi et al. conjectured that no convex loss can be calibrated with the PD in general [11, Section 2.1] because it would provide a polynomial algorithm to solve an NP-hard problem. Our approach thus leads to a direct proof of this conjecture.

In the next section, we describe our framework for learning to rank. We then present in Section 3 the general framework of calibration of [20], and give a new characterization of calibration for the evaluation metrics we consider (Theorem 2), and the implications of the convexity of a surrogate loss. Our main result is proved in Section 4. Section 5 concludes the paper, and Section 6 is a technical part containing the full proof of Theorem 2.

**Notation**   Let $\mathcal{V}$, $\mathcal{W}$ be two sets. A *set-valued* function $g$ from $\mathcal{V}$ to $\mathcal{W}$ maps all $v \in \mathcal{V}$ to a subset of $\mathcal{W}$ (set-valued functions appear in the paper as the result of $\arg\min$ operations). Given a subset $V$ of $\mathcal{V}$, the image of $V$ by $g$, denoted by $g(V)$, is the union of the images by $g$ of all members of $V$, i.e. $g(V) = \bigcup_{v \in V} g(v)$. If $n$ is a positive integer, $[n]$ is the set $\{1, ..., n\}$, and $\mathfrak{S}_n$ is the set of permutations of $[n]$. Boldface characters are used for vectors of $\mathbb{R}^n$. If $\mathbf{x} \in \mathbb{R}^n$, the $i$-th component of $\mathbf{x}$ is denoted by $x_i$ (normal font and subscript). The cardinal of a finite set $\mathcal{V}$ is denoted by $|\mathcal{V}|$.

## 2   Ranking Framework

We describe in this section the formal framework of ranking we consider. We first present the prediction problem we address, and then define the two main objects of our study: evaluation metrics for ranking and surrogate losses. We end the section with an outline of our technical contributions.

### 2.1   Framework and Definitions

We consider a framework similar to label ranking [9] or subset ranking [8]. Let $\mathcal{X}$ be a measurable space (the instance space). An instance $x \in \mathcal{X}$ represents a query and its associated $n$ items to rank, for an integer $n \geq 3$. The items are indexed from $1$ to $n$, and the goal is to order the set of item indexes $[n] = \{1, ..., n\}$ given $x$. The ordering (or ranking) is predicted by a *scoring function*, which is a measurable function $\mathbf{f} : \mathcal{X} \to \mathbb{R}^n$. For any input instance $x$, the scoring function $\mathbf{f}$ predicts a vector of $n$ relevance scores (one score for each item) and the ranking is predicted by sorting the item indexes by decreasing scores. We use permutations over $[n]$ to represent rankings, with the following conventions. First, given a permutation $\sigma$ in $\mathfrak{S}_n$, $k$ in $[n]$ is the rank of the item $\sigma(k)$; second, low ranks are better, so that $\sigma(1)$ is the top-ranked item.

Table 1: Formulas of $r(y, \sigma)$ for some common ranking evaluation metrics

| TYPE OF FEEDBACK | METRIC | FORMULA |
|---|---|---|
| $y \in \mathcal{Y} = \{0, ..., Y\}^n$, $Y \in \mathbb{N}, Y \geq 1$ | Discounted Cumulative Gain (higher values mean better performances) | $\sum_{k=1}^{n} \frac{2^{y_{\sigma(k)}} - 1}{\log(1+k)}$ |
| | Expected Reciprocal Rank (higher values mean better performances) | $\sum_{k=1}^{n} \frac{R_k}{k} \prod_{q=1}^{k-1} (1 - R_q), R_k = \frac{2^{y_{\sigma(k)}} - 1}{2^Y}$ |
| $y \in \mathcal{Y} = \{0, 1\}^n$ | Average Precision (higher values mean better performances) | $\frac{1}{|\{i : y_i = 1\}|} \sum_{i : y_i = 1} \sum_{k=1}^{\sigma^{-1}(i)} \frac{y_{\sigma(k)}}{\sigma^{-1}(i)}$ |
| $y \in \mathcal{Y} =$ all DAGs over $[n]$ | Pairwise Disagreement (lower values mean better performances) | $\sum_{i \to j \in y} I\left(\sigma^{-1}(i) > \sigma^{-1}(j)\right)$ |

The quality of a ranking is measured by a *ranking evaluation metric*, relatively to a *feedback*. The feedback space, denoted by $\mathcal{Y}$, is a *finite* set, and an evaluation metric is a function $r : \mathcal{Y} \times \mathfrak{S}_n \to \mathbb{R}$. We use the convention that lower values of $r$ are preferable, and thus when we discuss existing metrics for which higher values are better (e.g. the DCG, the AP or the ERR), we implicitly consider their opposite. Table 1 gives the formula and feedback spaces of the evaluation metrics that we discuss in the paper. The first three metrics – the DCG, the ERR and the AP – are commonly used for search engine evaluation. The feedback they consider is a vector of relevance judgments (one judgment per item). The last measure we consider is the PD, which is widely used when learning from pairwise preferences. For the feedback space of the PD, we follow [11] and take $\mathcal{Y}$ as the set of all directed acyclic graph (DAG) over $[n]$. For a DAG $y \in \mathcal{Y}$, there is an edge from item $i$ to $j$ (denoted $i \to j \in y$) when $i$ is preferred to $j$, or, equivalently when $i$ should have better rank than $j$.

In general, using a sorting algorithm, any ranking evaluation metric $r$ induces a quality measure on vectors of scores instead of rankings, considering that the sorting algorithm break ties randomly. Thus, using the following set-valued function from $\mathbb{R}^n$ to $\mathfrak{S}_n$, called $\arg \text{sort}$, which gives the set of rankings induced by a vector of scores:

$$\forall \mathbf{s} = (s_1, ..., s_n) \in \mathbb{R}^n, \quad \arg\text{sort}(\mathbf{s}) = \left\{ \sigma \in \mathfrak{S}_n | \forall k \in [n-1], s_{\sigma(k)} \geq s_{\sigma(k+1)} \right\},$$

the evaluation metric on vectors of scores induced by $r$ is defined by:

$$\forall y \in \mathcal{Y}, \forall \mathbf{s} \in \mathbb{R}^n, \quad r'(y, \mathbf{s}) = \sum_{\sigma \in \arg\text{sort}(\mathbf{s})} \frac{r(y, \sigma)}{|\arg\text{sort}(\mathbf{s})|} .$$

For a fixed, but unknown, probability measure $D$ on $\mathcal{X} \times \mathcal{Y}$, the objective of a learning algorithm is to find a scoring function $\mathbf{f}$ with low *ranking risk* $\mathcal{R}'(D, \mathbf{f}) = \int_{\mathcal{X} \times \mathcal{Y}} r'(y, \mathbf{f}(x)) \mathrm{d}D(\mathbf{x}, y)$ using a training set of (instance, feedback) pairs (e.g. drawn i.i.d. according to $D$).

The optimization of the empirical ranking risk is usually intractable because the ranking loss is discontinuous. To address this issue, algorithms optimize the empirical risk associated to a *surrogate loss* instead. Throughout the paper, we assume that this loss is *bounded below*, so that all the infima we take are well-defined. Without loss of generality, we assume that the surrogate loss has non-negative values, and we define a surrogate loss as a measurable function $\ell : \mathcal{Y} \times \mathbb{R}^n \to \mathbb{R}_+$. The surrogate risk of a scoring function $\mathbf{f}$ is then defined by $\mathcal{L}(D, \mathbf{f}) = \int_{\mathcal{X} \times \mathcal{Y}} \ell(y, \mathbf{f}(x)) \mathrm{d}D(x, y)$.

## 2.2 Outline of the Analysis

Any learning algorithm that performs empirical or structural risk minimization on the surrogate risk can, at most, be expected to reach low values of the surrogate risk. The question we address in this paper is whether such an algorithm provably solves the real learning task, which is to achieve low values of the ranking risk. More formally, the criterion under study is whether the following implication holds for every sequence of scoring functions $(\mathbf{f}_k)_{k \geq 0}$ and every data distribution $D$:

$$\mathcal{L}(D, \mathbf{f}_k) \underset{k \to \infty}{\longrightarrow} \inf_{\mathbf{f}} \mathcal{L}(D, \mathbf{f}) \quad \Rightarrow \quad \mathcal{R}'(D, \mathbf{f}_k) \underset{k \to \infty}{\longrightarrow} \inf_{\mathbf{f}} \mathcal{R}'(D, \mathbf{f}) \tag{1}$$

where the infima are taken over all scoring functions. In particular, we show that if a surrogate loss is convex in the sense that $\ell(y, .)$ is convex for every $y \in \mathcal{Y}$, and if the evaluation metric is the AP,

the ERR or the PD, then there are distributions and sequences of scoring functions for which (1) does not hold. In other words, we show that learning-to-rank algorithms that define their objective through a convex surrogate loss cannot provably optimize any of these evaluation metrics.

In order to perform a general analysis for all the three evaluation metrics, we consider Assumption **(A)** below, which formalizes the common property of these metrics that is relevant to our study. Intuitively, it means that for any given item, there is a feedback for which the performance only depends on the rank of this item, with a strict improvement of performances when one improves the rank of the item:

**(A)** $\quad \exists \beta_1 < \beta_2 < ... < \beta_n \quad$ such that $\quad \forall i \in [n], \exists y \in \mathcal{Y} : \forall \sigma \in \mathfrak{S}_n, r(y, \sigma) = \beta_{\sigma^{-1}(i)}.$

Note that in the assumption, the values of $\beta_k$ (i.e. the performance when item $i$ is predicted at rank $k$) are the same for all items. This is not a strong requirement because the metrics we consider do not depend on how we index the elements. The DCG, the AP and the ERR satisfy **(A)**: for each $i$, we take the vector of relevance with a 1 for item $i$ and 0 for all other items so that the values of the metrics only depends on the rank of $i$ (which should be ranked first). The PD satisfies Assumption **(A)** as well: for each $i$, take $y$ as the DAG containing the edges $i \to j, \forall j \in [n] \setminus \{i\}$ and only those edges. For this feedback, $i$ is preferred to all other items (and no preference is specified regarding the other items) and thus the quality of a ranking only depends on the rank of $i$.

Our analysis is organized as follows. In the next section, we introduce the notion of a calibrated surrogate loss defined by Steinwart [20], which is a criterion equivalent to (1). We then obtain a new condition that is equivalent to calibration when **(A)** holds, and finally we restrict our attention to evaluation metrics satisfying **(A)** and to convex surrogate losses. In that context, using our new condition for calibration, we show that evaluation metrics with a calibrated surrogate loss necessarily satisfy a specific property. Then, in Section 4, we prove that the AP, the ERR and the PD do not satisfy this property. Since Assumption **(A)** holds for these three metrics, this latter result implies that they do not have any convex and calibrated surrogate loss. Equivalently, it implies that (1) does not hold in general for these metrics if the surrogate loss is convex.

## 3 A New Characterization of Calibration

We present in this section the notion of calibration as studied in [20], which is the basis of our work. Then, we provide a characterization of calibration more specific to the evaluation metrics we consider, that relates more closely calibrated surrogate losses and evaluation metrics. This more specific characterization of calibration is the starting point of the analysis of convex and calibrated surrogate losses carried out in the last subsection and that allows us to state the results of Section 4.

### 3.1 The Framework of Calibration

Applying the general results of [20] to our setting, the criterion defined by (1) can be studied by restricting our attention to the contributions of a single instance to the surrogate and ranking risk. These contributions are called the *inner surrogate risk* and the *inner ranking risk* respectively. Denoting the set of probability distributions over $\mathcal{Y}$ by $\mathcal{P} = \left\{ p : \mathcal{Y} \to [0, 1] | \sum_{y \in \mathcal{Y}} p(y) = 1 \right\}$, the inner risks are respectively defined for all $p \in \mathcal{P}$ and all $\mathbf{s} \in \mathbb{R}^n$ by:

$$L(p, \mathbf{s}) = \sum_{y \in \mathcal{Y}} p(y) \ell(y, \mathbf{s})$$

and $\quad R'(p, \mathbf{s}) = \sum_{\sigma \in \arg \operatorname{sort}(\mathbf{s})} \frac{R(p, \sigma)}{|\arg \operatorname{sort}(\mathbf{s})|}, \quad$ where $\quad \forall \sigma \in \mathfrak{S}_n, R(p, \sigma) = \sum_{y \in \mathcal{Y}} p(y) r(y, \sigma).$

Their optimal values are denoted by $\underline{L}(p) = \inf_{\mathbf{s} \in \mathbb{R}^n} L(p, \mathbf{s})$ and $\underline{R}'(p) = \underline{R}(p) = \min_{\sigma \in \mathfrak{S}_n} R(p, \sigma).$

More precisely, [20, Theorem 2.8] shows that (1) holds for any distribution $D$ and any sequence of scoring functions if and only if the surrogate loss is $r$-calibrated according to the definition below. Similarly to (1), the calibration is an implication of two limits, but it involves the inner risks $L$ and $R'$ instead of the risks $\mathcal{L}$ and $\mathcal{R}'$. For convenience in the rest of the work, we write the implication

between the two limits of $L$ and $R'$ as an inclusion of the sets of near-optimal vectors of scores. For any $\varepsilon > 0$ and $\delta > 0$, the latter sets are respectively denoted by

$$\mathcal{M}_\ell(p, \delta) = \{\mathbf{s} \in \mathbb{R}^n | L(p, \mathbf{s}) - \underline{L}(p) < \delta\} \text{ and } \mathcal{M}_r(p, \varepsilon) = \{\mathbf{s} \in \mathbb{R}^n | R'(p, \mathbf{s}) - \underline{R}'(p) < \varepsilon\} ,$$

so that the definition of an $r$-calibrated loss is the following:

**Definition 1.** *[20, Definition 2.7] The surrogate loss $\ell$ is $r$-calibrated if*

$$\forall p \in \mathcal{P}, \forall \varepsilon > 0, \; \exists \delta > 0 : \mathcal{M}_\ell(p, \delta) \subseteq \mathcal{M}_r(p, \varepsilon) .$$

## 3.2 Calibration through Optimal Rankings

Definition 1 is the starting point of our analysis, and our goal is to show that if the evaluation metric is the AP, the ERR or the PD, then no convex surrogate loss can satisfy it. The goal of this subsection is to give a stronger characterization of $r$-calibrated surrogate losses when Assumption **(A)** holds. The starting point of this characterization is to rewrite Definition 1 in terms of rankings induced by the sets of near-optimal scores, from which we can deduce that $\ell$ is $r$-calibrated if and only if[1]:

$$\forall p \in \mathcal{P}, \forall \varepsilon > 0, \; \exists \delta > 0 : \arg\mathrm{sort}(\mathcal{M}_\ell(p, \delta)) \subseteq \arg\mathrm{sort}(\mathcal{M}_r(p, \varepsilon)) .$$

In contrast to this characterization of calibration, our result (Theorem 2 below), which is specific to metrics that satisfy **(A)**, replaces the inclusion (which can be strict in general) of sets of ranking by an equality when $\varepsilon$ tends to 0. More specifically, we define the set of optimal rankings for the inner ranking risk with the following set-valued function from $\mathcal{P}$ to $\mathfrak{S}_n$:

$$\forall p \in \mathcal{P}, \quad \mathcal{A}_r(p) = \arg\min_{\sigma \in \mathfrak{S}_n} R(p, \sigma) ,$$

so that when Assumption **(A)** holds, the set of optimal rankings is *equal* to a set of rankings induced by near-optimal scores of the inner surrogate risk:

**Theorem 2.** *If Assumption **(A)** holds, then $\ell$ is $r$-calibrated if and only if*

$$\forall p \in \mathcal{P}, \exists \delta > 0 \text{ s.t. } \arg\mathrm{sort}(\mathcal{M}_\ell(p, \delta)) = \mathcal{A}_r(p) .$$

The proof of Theorem 2 is deferred to Section 6 at the end of the paper. This theorem enables us to relate the surrogate loss and the evaluation metric so that the convexity of $\ell$ induces some constraints on $r$ that are not satisfied by all evaluation metrics.

## 3.3 The implication of Convexity on Sets of Optimal Rankings

If $\ell(y, .)$ is convex for all $y \in \mathcal{P}$, then the inner risk $L(p, .)$ is also convex for every distribution $p \in \mathcal{P}$. This implies that $\mathcal{M}_\ell(p, \delta)$ is a convex subset of $\mathbb{R}^n$. Thus, if $\ell$ is $r$-calibrated, then Theorem 2 implies that $\mathcal{A}_r(p) = \arg\mathrm{sort}(\mathcal{M}_\ell(p, \delta))$ is a set of rankings induced by a convex set of $\mathbb{R}^n$.

The following theorem presents a condition that the set $\mathcal{A}_r(p)$ must satisfy if it is generated by a convex set of scores: if there exists at least one pair of items $(i, j)$ which are inverted in two rankings of $\mathcal{A}_r(p)$, then $i$ and $j$ are "indifferent" in $\mathcal{A}_r(p)$:

**Theorem 3.** *Assume that for all $y \in \mathcal{Y}$, the function $\mathbf{s} \mapsto \ell(y, \mathbf{s})$ is convex. If Assumption **(A)** holds and $\ell$ is $r$-calibrated, then $r$ satisfies: $\forall p \in \mathcal{P}, \forall i, j \in [n], \forall \sigma, \sigma' \in \mathcal{A}_r(p)$,*

$$\sigma^{-1}(i) < \sigma^{-1}(j) \text{ and } \sigma'^{-1}(i) > \sigma'^{-1}(j) \; \Rightarrow \; \exists \mathbf{s} \in \mathbb{R}^n : s_i = s_j \text{ and } \arg\mathrm{sort}(\mathbf{s}) \subseteq \mathcal{A}_r(p) . \quad (2)$$

*Proof of Theorem 3.* Assume that the conditions of the theorem are satisfied. From now on, we fix some $p \in \mathcal{P}$ and two $i$ and $j$ in $[n]$. Take $\sigma$ and $\sigma'$ in $\mathcal{A}_r(p)$ and assume that $\sigma^{-1}(i) < \sigma^{-1}(j)$ and $\sigma'^{-1}(i) > \sigma'^{-1}(j)$. Since Assumption **(A)** holds, there is a $\delta > 0$ such that $\mathcal{A}_r(p) = \arg\mathrm{sort}(\mathcal{M}_\ell(p, \delta))$ by Theorem 2. Thus, there are two score vectors $\mathbf{u}$ and $\mathbf{v}$ in $\mathcal{M}_\ell(p, \delta)$ such that $u_i \geq u_j$ ($\mathbf{u}$ induces the ranking $\sigma$) and $v_i \leq v_j$ ($\mathbf{v}$ induces the ranking $\sigma'$).

Moreover, since $\ell$ is convex, the function $L(p, .)$ is convex for every $p \in \mathcal{P}$, and thus $\mathcal{M}_\ell(p, \delta)$ is convex. Consequently, for all $t \in [0, 1]$, the vector $\boldsymbol{\gamma}(t) = (1 - t)\mathbf{u} + t\mathbf{v}$ belongs to $\mathcal{M}_\ell(p, \delta)$. We define $g : t \mapsto \gamma_i(t) - \gamma_j(t)$ for $t \in [0, 1]$. Then, $g$ is continuous, with $g(0) = u_i - u_j \geq 0$ and $g(1) = v_i - v_j \leq 0$. By the intermediate value theorem, there is $t_0 \in [0, 1]$ such that $g(t_0) = 0$. The consequence is that the score vector $\mathbf{s}$, defined by $\mathbf{s} = \boldsymbol{\gamma}(t_0)$, satisfies $\mathbf{s} \in \mathcal{M}_\ell(p, \delta)$ and $s_i = s_j$. $\quad\square$

Table 2: Examples for Corollary 4. There are three elements to rank. $i \succ j \succ k$ represents the permutation that ranks item $i$ first, $j$ second and $k$ last. For the ERR and the AP, we consider binary relevance judgments. $p_{110}$ denotes a Dirac distribution at the feedback vector $y = [1, 1, 0]$. $p_{001}$ is defined similarly. For the Pairwise Disagreement, $p_{1 \succ 2 \succ 3}$ is the Dirac distribution at the DAG containing the edges $1 \to 2$, $2 \to 3$ and $1 \to 3$, i.e. the DAG corresponding to $1 \succ 2 \succ 3$. The Dirac distribution at the DAG containing only the edge $3 \to 1$ is denoted by $p_{3 \succ 1}$. In all cases, $\tilde{p}(\alpha)$ is a mixture between two Dirac distributions. The sets $\mathcal{A}_r(\tilde{p}(\alpha))$ are obtained by direct calculations. The set $\mathcal{A}_r(\tilde{p}(\alpha))$ is the same for all $\alpha$s in the range given in the third column.

| DISTRIBUTION $\tilde{p}(\alpha)$ | METRIC | RANGE OF $\alpha$ | $\mathcal{A}_r(\tilde{p}(\alpha))$ |
|---|---|---|---|
| $(1-\alpha)p_{110} + \alpha p_{001}$ | ERR | $\alpha \in \left(\frac{1}{3}, \frac{1}{2}\right)$ | $\{(1 \succ 3 \succ 2), (2 \succ 3 \succ 1)\}$ |
| | AP | $\alpha = \frac{5}{13}$ | $\{(1 \succ 2 \succ 3), (3 \succ 1 \succ 2),$ $(2 \succ 1 \succ 3), (3 \succ 2 \succ 1)\}$ |
| $(1-\alpha)p_{1 \succ 2 \succ 3} + \alpha p_{3 \succ 1}$ | PD | $\alpha \in \left(\frac{2}{3}, 1\right)$ | $\{(2 \succ 3 \succ 1), (3 \succ 1 \succ 2)\}$ |

The contrapositive of Theorem 3 is our technical tool to prove the nonexistence of convex and calibrated losses. Indeed, for a given evaluation metric $r$, if we are able to exhibit a distribution $p \in \mathcal{P}$ such that (2) is not satisfied, this evaluation metric cannot have a surrogate loss both convex and calibrated. In the next subsection, we apply this argument to the AP, the ERR and the PD.

**Remark 1.** *It has been proved by several authors that there exist convex surrogate losses that are DCG-calibrated [8, 2, 17]. Thus, the DCG satisfies (2). It can be seen by observing that the optimal rankings for the DCG are* exactly *those generated by sorting the items according to the vector of score* $\mathbf{s}^*(p)$ *defined by* $s_i^*(p) = \sum_{y \in \mathcal{Y}} p(y)2^{y_i}$, *i.e.* $\mathcal{A}_r(p) = \arg \operatorname{sort}(\mathbf{s}^*(p))$.

## 4 Nonexistence Results

We now present the main result of the nonexistence of convex, calibrated surrogate losses:

**Corollary 4.** *No convex surrogate loss is calibrated with respect to the AP, the ERR or the PD.*

*Proof.* We consider the case where there are three elements to rank, and we use the examples and the notations of Table 2. Since all three metrics satisfy **(A)**, Theorem 3 implies that if $r$ (taken as either the AP, the ERR or the PD) has a calibrated, convex surrogate loss, then, for any distribution $\tilde{p}(\alpha)$, we have: if item $i$ is preferred to $j$ according to a ranking in $\mathcal{A}_r(\tilde{p}(\alpha))$, and $j$ is preferred to $i$ according to another ranking in $\mathcal{A}_r(\tilde{p}(\alpha))$, then one of the two assertions below must hold:

(a) $\{(i \succ j \succ k), (j \succ i \succ k)\} \subseteq \mathcal{A}_r(\tilde{p}(\alpha))$, (b) $\{(k \succ i \succ j), (k \succ j \succ i)\} \subseteq \mathcal{A}_r(\tilde{p}(\alpha))$

because there exists $\mathbf{s} \in \mathbb{R}^3$ such that $\arg \operatorname{sort}(\mathbf{s}) \subseteq \mathcal{A}_r(\tilde{p}(\alpha))$ for which either $s_i = s_j \leq s_k$ or $s_i = s_j \geq s_k$. Now, let us consider the case of the ERR. Taking an arbitrary $\alpha \in \left(\frac{1}{3}, \frac{1}{2}\right)$, we see on the last column of Table 2 that $\mathcal{A}_r(\tilde{p}(\alpha))$ contains two rankings: one of them ranks item 1 before item 2, and the other one ranks 2 before 1. If the ERR had a convex calibrated surrogate loss, then either (a) or (b) should hold. However, we see that neither (a) nor (b) holds. Thus their is no convex, ERR-calibrated surrogate loss. For the AP, a similar argument with items 1 and 3 leads to the conclusion. For the PD, taking any two items leads to the result. □

A first consequence of Corollary 4 is that for ranking problems evaluated in terms of AP, ERR or PD, surrogate losses defined as convex upper bounds on an evaluation metric as discussed in [24], as well as convex surrogate losses proposed in the structured output framework such as $\text{SVM}^{map}$ [25] are not calibrated with respect to the evaluation metric they are designed for. The convex surrogate losses used by most participants of the recent Yahoo! Learning to Rank Challenge [6] are also not calibrated with respect to the ERR, the official evaluation metric of the challenge. The fact that the minimization of a non-calibrated surrogate risk leads to suboptimal prediction functions on some data distributions suggests that convex losses are not a definitive solution to learning to rank. Significant improvements in performances may then be obtained by switching to other approaches than the optimization of a convex risk.

# 5 Conclusion

We proved that convex surrogate losses cannot be calibrated with three major ranking evaluation metrics. The result cast light on the intrinsic limitations of all algorithms based on (empirical) convex risk minimization for ranking, even though most existing algorithms for learning to rank follow this approach. A possible direction for future work is to study whether the calibration of convex losses can be obtained under low noise conditions. Such studies was carried out for the PD [11], and calibrated, convex surrogate losses were found for special cases of practical interest. Nonetheless, in order to obtain algorithms that do not rely on low noise assumptions, our results suggest to explore whether alternatives to convex surrogate approaches can lead to improvements in terms of performances. A first possibility is to turn to non-convex losses for ranking as in [10, 3], and to study the calibration of such losses. Another alternative is to use another surrogate approach than scoring, such as directly learning pairwise preferences [13], even though the reconstruction of an optimal ranking, given the pairwise predictions, that is optimal for evaluation metrics such as the AP, the ERR or the PD is still mostly an open issue.

# 6 Proof of Theorem 2

We remind the statement of Theorem 2: if $r$ satisfies **(A)**, then $\ell$ is $r$-calibrated if and only if for all $p \in \mathcal{P}$, there exists $\delta > 0$ such that $\mathcal{A}_r(p) = \arg\mathrm{sort}(\mathcal{M}_\ell(p, \delta))$. We prove the result using the following set-valued function which defines the set of optimal rankings for the inner surrogate risk:

$$\mathcal{A}_\ell(p) = \arg\min_{\sigma \in \mathfrak{S}_n} \widetilde{L}(p, \sigma) \;\; \text{where} \;\; \widetilde{L}(p, \sigma) = \inf\left\{ L(p, \mathbf{s}) \,|\, \mathbf{s} \in \mathbb{R}^n \text{ s.t. } \sigma \in \arg\mathrm{sort}(\mathbf{s}) \right\}.$$

Then, Theorem 2 is a direct implication of the two following claims that we prove in this section:

**(a)** the assertion $\forall p \in \mathcal{P}, \exists \delta > 0, \arg\mathrm{sort}(\mathcal{M}_\ell(p, \delta)) = \mathcal{A}_\ell(p)$ is true in general;

**(b)** if Assumption **(A)** holds, then $\ell$ is $r$-calibrated if and only if $\forall p \in \mathcal{P}, \mathcal{A}_\ell(p) = \mathcal{A}_r(p)$.

The proof of these two claims is based on three lemmas that we present before the final proof. The first lemma, which does not need any assumption on the evaluation metric, both proves equality **(a)** and provides a general characterization of calibration in terms of optimal rankings. The second lemma concerns the surrogate loss; it states that a slight perturbation in $p$ does not affect "too much" $\mathcal{A}_\ell(p)$. The third lemma concerns evaluation metrics and gives a simple consequence of Assumption **(A)**. The final proof of Theorem 2 connects all these pieces together to prove **(b)**.

**Lemma 5.** *The following claims are true:*

*(i)* $\forall p \in \mathcal{P}, \forall \delta > 0, \quad \mathcal{A}_\ell(p) \subseteq \arg\mathrm{sort}(\mathcal{M}_\ell(p, \delta))$.

*(ii)* $\forall p \in \mathcal{P}, \exists \delta_0 > 0 : \mathcal{A}_\ell(p) = \arg\mathrm{sort}(\mathcal{M}_\ell(p, \delta_0))$.

*(iii)* $\ell$ *is* $r$*-calibrated if and only if:* $\forall p \in \mathcal{P}, \mathcal{A}_\ell(p) \subseteq \mathcal{A}_r(p)$.

*Proof.* *(i)* Fix $p \in \mathcal{P}$ and $\delta > 0$. Let $\sigma \in \mathcal{A}_\ell(p)$. By the definition of $\widetilde{L}$, there is an $\mathbf{s} \in \mathbb{R}^n$ such that $\sigma \in \arg\mathrm{sort}(\mathbf{s})$ and $L(p, \mathbf{s}) - \widetilde{L}(p, \sigma) < \delta$. Since $\widetilde{L}(p, \sigma) = \min_{\sigma' \in \mathfrak{S}_n} \widetilde{L}(p, \sigma') = \underline{L}(p)$, we have $L(p, \mathbf{s}) - \underline{L}(p) < \delta$. This proves $\mathbf{s} \in \mathcal{M}_\ell(p, \delta)$ and thus $\sigma \in \arg\mathrm{sort}(\mathcal{M}_\ell(p, \delta))$.

*(ii)* Fix $p \in \mathcal{P}$ and take $\delta_0 = \min_{\sigma \notin \mathcal{A}_\ell(p)} \widetilde{L}(p, \sigma) - \underline{L}(p) > 0$, with the convention $\min \emptyset = +\infty$. The choice of $\delta_0$ guarantees that $\forall \mathbf{s} \in \mathbb{R}^n, L(p, \mathbf{s}) - \underline{L}(p) < \delta_0 \Rightarrow \arg\mathrm{sort}(\mathbf{s}) \subseteq \mathcal{A}_\ell(p)$, which is equivalent to $\arg\mathrm{sort}(\mathcal{M}_\ell(p, \delta_0)) \subseteq \mathcal{A}_\ell(p)$. The reverse inclusion is given by the first point.

*(iii)* Since $r$ can only take a finite set of values, we can prove that $\ell$ is $r$-calibrated if and only if: $\forall p \in \mathcal{P}, \exists \delta > 0 : \forall \mathbf{s} \in \mathbb{R}^n, L(p, \mathbf{s}) - \underline{L}(p) < \delta \Rightarrow R'(p, \mathbf{s}) = \underline{R}'(p)$. Moreover, we have $R'(p, \mathbf{s}) = \underline{R}'(p) \Leftrightarrow \arg\mathrm{sort}(\mathbf{s}) \subseteq \mathcal{A}_r(p)$ since $R'(p, \mathbf{s})$ is the mean of $R(p, \sigma)$ for $\sigma \in \arg\mathrm{sort}(\mathbf{s})$. Thus, $\ell$ is $r$-calibrated if and only if for every $p \in \mathcal{P}$, there exists $\delta > 0$ such that $\arg\mathrm{sort}(\mathcal{M}_\ell(p, \delta)) \subseteq \mathcal{A}_r(p)$. This characterization and the first two points give the result. $\square$

We now present a more technical result on $\mathcal{A}_\ell$, which shows the set of optimal rankings cannot dramatically change under a slight perturbation in the distribution over the feedback space. From now on, for any $p \in \mathcal{P}$ and any $\eta > 0$, we denote by $B(p, \eta)$ the open ball of $\mathcal{P}$ (with respect to $\|.\|_1$) of radius $\eta$ centered at $p$, i.e. $B(p, \eta) = \{p' \in \mathcal{P} | \|p - p'\|_1 < \eta\}$.

**Lemma 6.** $\forall p \in \mathcal{P}, \exists \eta > 0$ such that $\mathcal{A}_\ell(B(p, \eta)) = \mathcal{A}_\ell(p)$.

*Proof.* Note that $\mathcal{A}_\ell(p) \subseteq \mathcal{A}_\ell(B(p, \eta))$ since $p \in B(p, \eta)$. We now prove $\mathcal{A}_\ell(B(p, \eta)) \subseteq \mathcal{A}_\ell(p)$; the main argument is that $\widetilde{L}(., \sigma)$ is continuous for every $\sigma$ because $\mathcal{Y}$ is finite [23, Theorem 2]. Indeed, let us fix $p \in \mathcal{P}$ and define $\varepsilon = \frac{1}{2} \left( \min_{\sigma' \notin \mathcal{A}_\ell(p)} \widetilde{L}(p, \sigma') - \underline{L}(p) \right)$. For each $\sigma \in \mathfrak{S}_n$, since $\widetilde{L}(., \sigma)$ is continuous, there exists $\eta_\sigma > 0$ such that $\forall p' \in B(p, \eta_\sigma), |\widetilde{L}(p', \sigma) - \widetilde{L}(p, \sigma)| < \varepsilon$.

Let $\eta = \min_{\sigma \in \mathfrak{S}_n} \eta_\sigma$, and let $p'$ be an arbitrary member of $B(p, \eta)$. By the definition of $\varepsilon$, we have:

$$\forall \sigma' \notin \mathcal{A}_\ell(p), \widetilde{L}(p', \sigma') = \widetilde{L}(p', \sigma') - \widetilde{L}(p, \sigma') + \widetilde{L}(p, \sigma') - \underline{L}(p) + \underline{L}(p) > -\varepsilon + 2\varepsilon + \underline{L}(p).$$

Thus, $\forall \sigma' \notin \mathcal{A}_\ell(p), \widetilde{L}(p', \sigma') > \underline{L}(p) + \varepsilon$. Additionally, the definition of $\eta$ gives $\forall \sigma \in \mathcal{A}_\ell(p), \widetilde{L}(p', \sigma) < \underline{L}(p) + \varepsilon$. Thus, we have $\min_{\sigma' \notin \mathcal{A}_\ell(p)} \widetilde{L}(p', \sigma') > \min_{\sigma \in \mathcal{A}_\ell(p)} \widetilde{L}(p', \sigma)$. This proves that a ranking that is not optimal for $\widetilde{L}(p, .)$ cannot be optimal for $\widetilde{L}(p', .)$. Thus $\mathcal{A}_\ell(p') \subseteq \mathcal{A}_\ell(p)$ from which we conclude $\mathcal{A}_\ell(B(p, \eta)) \subseteq \mathcal{A}_\ell(p)$. $\square$

Now that we have studied the properties of $\mathcal{A}_\ell$, we analyze in more depth the evaluation metrics. We prove the following consequence of Assumption **(A)**: for each possible ranking there is a distribution over the feedback space for which this ranking is the unique optimal ranking.

**Lemma 7.** If Assumption **(A)** holds, then $\forall \sigma \in \mathfrak{S}_n, \exists p^\sigma \in \mathcal{P}$ such that $\mathcal{A}_r(p^\sigma) = \{\sigma\}$.

*Proof.* Assume **(A)** holds, and, for each item $k$, let us denote by $y^k$ the feedback corresponding to item $k$ in Assumption **(A)**. Now, let us take some $\sigma \in \mathfrak{S}_n$ and define $p^\sigma$ as $p^\sigma(y^k) = \alpha_{\sigma^{-1}(k)}$ with $\alpha_1 > ... > \alpha_n > 0$ and $\sum_{k=1}^{n} \alpha_k = 1$. Then, for any $\sigma' \in \mathfrak{S}_n$, we have the equality $R(p^\sigma, \sigma') = \sum_{k=1}^{n} \alpha_{\sigma^{-1}(k)} r(y^k, \sigma') = \sum_{k=1}^{n} \alpha_{\sigma^{-1}(k)} \beta_{\sigma'^{-1}(k)}$. Since the $\alpha$s are non-negative, and since there are ties neither the $\alpha$s nor in the $\beta$s, the rearrangement inequality implies that the minimum value of $R(p^\sigma, \sigma')$ is obtained for the single permutation $\sigma'$ for which the $\beta_{\sigma'^{-1}(k)}$ are in reverse order relatively to the $\alpha_{\sigma^{-1}(k)}$ (i.e. smaller values $\beta_{\sigma'^{-1}(k)}$ should be associated to greater values of $\alpha_{\sigma^{-1}(k)}$). Since the $\alpha_k$s are decreasing with $k$ and the $\beta_k$s are increasing, the minimum value of $\sigma' \mapsto R(p^\sigma, \sigma') = \sum_{k=1}^{n} \alpha_{\sigma^{-1}(k)} \beta_{\sigma'^{-1}(k)}$ is obtained if and only if $\sigma^{-1} = \sigma'^{-1}$ (i.e. $\sigma' = \sigma$). $\square$

*Proof of Theorem 2.* We remind to the reader that by the second point of Lemma 5, for any $p \in \mathcal{P}$, there is $\delta > 0$ such that $\mathcal{A}_\ell(p) = \arg\text{sort}(\mathcal{M}_\ell(p, \delta))$. What remains to show is that if Assumption **(A)** holds, then $\ell$ is $r$-calibrated if and only if $\forall p \in \mathcal{P}, \mathcal{A}_\ell(p) = \mathcal{A}_r(p)$.

*("if" direction)* If $\forall p \in \mathcal{P}, \mathcal{A}_\ell(p) = \mathcal{A}_r(p)$ then $\ell$ is $r$-calibrated by Lemma 5.

*("only if" direction)* Assume that **(A)** holds and that $\ell$ is $r$-calibrated. Let $p \in \mathcal{P}$. By Point *(iii)* of Lemma 5, we know that $\mathcal{A}_\ell(p) \subseteq \mathcal{A}_r(p)$. We now prove the reverse inclusion $\mathcal{A}_\ell(p') \subseteq \mathcal{A}_r(p')$.

By Lemma 6, there exists some $\eta > 0$ such that $\mathcal{A}_\ell(B(p, \eta)) = \mathcal{A}_\ell(p)$. Let $\sigma \in \mathcal{A}_r(p)$. The idea is to use Lemma 7 to find some $p' \in B(p, \eta)$ such that $\mathcal{A}_\ell(p') = \{\sigma\}$ which would prove $\sigma \in \mathcal{A}_\ell(p)$ and thus the result. The rest of the proof consists in building $p'$.

Using Lemma 7, let $p^\sigma \in \mathcal{P}$ such that $\mathcal{A}_r(p^\sigma) = \{\sigma\}$. Now, let $p' = (1 - \frac{\eta}{4})p + \frac{\eta}{4}p^\sigma$. Then, we have $\|p - p'\|_1 = \frac{\eta}{4}\|p - p^\sigma\|_1 \leq \eta/2$ and thus $p' \in B(p, \eta)$. Moreover, $\mathcal{A}_r(p') = \{\sigma\}$ since $\sigma$ is optimal for both $p$ and $p^\sigma$, and any other permutation is suboptimal for $p^\sigma$. We also have $\mathcal{A}_\ell(p') = \{\sigma\}$ because $\mathcal{A}_\ell$ has non-empty values and calibration implies that $\mathcal{A}_\ell(p') \subseteq \mathcal{A}_r(p')$ by Lemma 5. $\square$

### Acknowledgements

This work was partially funded by the French DGA. The authors thank M. R. Amini, D. Buffoni, S. Clémençon, L. Denoyer and G. Wisniewski for their comments and suggestions.

## Footnotes

[1] We remind to the reader the notation $\arg\mathrm{sort}(\mathcal{M}_\ell(p, \delta)) = \bigcup_{\mathbf{s} \in \mathcal{M}_\ell(p, \delta)} \arg\mathrm{sort}(\mathbf{s})$.

# References

[1] P. L. Bartlett, M. I. Jordan, and J. D. McAuliffe. Convexity, classification, and risk bounds. *J. of the American Stat. Assoc.*, pages 1–36, 2006.

[2] D. Buffoni, C. Calauzènes, P. Gallinari, and N. Usunier. Learning scoring functions with order-preserving losses and standardized supervision. In *Proc. of the Intl. Conf. on Mach. Learn.*, pages 825–832, 2011.

[3] C. J. Burges, R. Ragno, and Q. V. Le. Learning to rank with nonsmooth cost functions. In *Proc. of Adv. in Neural Info. Proc. Syst.*, pages 193–200, 2007.

[4] Y. Cao, J. Xu, T.-Y. Liu, H. Li, Y. Huang, and H.-W. Hon. Adapting ranking svm to document retrieval. In *Proc. of the ACM SIGIR Conf. on Res. and Dev. in Info. Retr.*, pages 186–193, 2006.

[5] A. Chang, C. Rudin, M. Cavaretta, R. Thomas, and G. Chou. How to reverse-engineer quality rankings. *Mach. Learn.*, 88(3):369–398, Sept. 2012.

[6] O. Chapelle and Y. Chang. Yahoo! learning to rank challenge overview. *J. of Mach. Learn. Res.*, 14:1–24, 2011.

[7] S. Clémençon, G. Lugosi, and N. Vayatis. Ranking and scoring using empirical risk minimization. In *Proc. of the 18th Conf. on Learning Theory*, COLT'05, pages 1–15, 2005.

[8] D. Cossock and T. Zhang. Statistical analysis of bayes optimal subset ranking. *IEEE Trans. Info. Theory*, 54:5140–5154, 2008.

[9] O. Dekel, C. D. Manning, and Y. Singer. Log-linear models for label ranking. In *Proc. of Advances in Neural Information Processing Systems (NIPS)*, 2003.

[10] C. B. Do, Q. Le, C. H. Teo, O. Chapelle, and A. Smola. Tighter bounds for structured estimation. In *Proc. of Adv. in Neural Inf. Processing Syst.*, pages 281–288, 2008.

[11] J. Duchi, L. W. Mackey, and M. I. Jordan. On the consistency of ranking algorithms. In *Proc. of the Int. Conf. on Mach. Learn.*, pages 327–334, 2010.

[12] Y. Freund, R. Iyer, R. E. Schapire, and Y. Singer. An efficient boosting algorithm for combining preferences. *J. of Mach. Learn. Res.*, 4:933–969, 2003.

[13] E. Hullermeier, J. Furnkranz, W. Cheng, and K. Brinker. Label ranking by learning pairwise preferences. *Artificial Intelligence*, 172(16-17):1897–1916, Nov. 2008.

[14] T. Joachims. Optimizing search engines using clickthrough data. In *Proc. of Know. Disc. and Dat. Mining (SIGKDD)*, pages 133–142, 2002.

[15] W. Kotlowski, K. Dembczynski, and E. Huellermeier. Bipartite ranking through minimization of univariate loss. In *Proc. of the Intl. Conf. on Mach. Learn.*, pages 1113–1120, 2011.

[16] T.-Y. Liu. Learning to rank for information retrieval. *Foundations and Trends in Information Retrieval*, 3:225–331, March 2009.

[17] P. D. Ravikumar, A. Tewari, and E. Yang. On ndcg consistency of listwise ranking methods. *J. of Mach. Learn. Res. - Proc. Track*, 15:618–626, 2011.

[18] M. D. Reid and R. C. Williamson. Surrogate Regret Bounds for Proper Losses. In *Proc. of the Intl. Conf. on Mach. Learn.*, pages 897–904, 2009.

[19] C. Scott. Surrogate losses and regret bounds for cost-sensitive classification with example-dependent costs. *Proc. of the Intl. Conf. on Mach. Learn.*, pages 153–160, 2011.

[20] I. Steinwart. How to compare different loss functions and their risks. *Constructive Approximation*, 26(2):225–287, 2007.

[21] M. Taylor, J. Guiver, S. Robertson, and T. Minka. Softrank: optimizing non-smooth rank metrics. In *Proceedings of the international conference on Web search and web data mining*, WSDM '08, pages 77–86, 2008.

[22] E. Voorhees, D. Harman, N. I. of Standards, and T. (U.S.). *TREC: experiment and evaluation in information retrieval*. Digital libraries and electronic publishing. MIT Press, 2005.

[23] R. A. Wijsman. Continuity of the bayes risk. *The Annals of Math. Stat.*, 41(3):pp. 1083–1085, 1970.

[24] J. Xu, T.-Y. Liu, M. Lu, H. Li, and W.-Y. Ma. Directly optimizing evaluation measures in learning to rank. In *Proceedings of the 31st annual international ACM SIGIR conference on Research and development in information retrieval*, SIGIR '08, pages 107–114, 2008.

[25] Y. Yue, T. Finley, F. Radlinski, and T. Joachims. A support vector method for optimizing average precision. In *Proc. of the ACM SIGIR Intl. Conf. on Res. and Dev. in Info. Retr.*, pages 271–278, 2007.

[26] T. Zhang. Statistical analysis of some multi-category large margin classification methods. *J. of Mach. Learn. Res.*, 5:1225–1251, 2004.

[27] T. Zhang. Statistical behavior and consistency of classification methods based on convex risk minimization. *The Annals of Stat.*, 32(1):pp. 56–85, 2004.

